# Autonomous helicopter flight
# via Reinforcement Learning

**Andrew Y. Ng**
Stanford University
Stanford, CA 94305

**H. Jin Kim, Michael I. Jordan, and Shankar Sastry**
University of California
Berkeley, CA 94720

## Abstract

Autonomous helicopter flight represents a challenging control problem, with complex, noisy, dynamics. In this paper, we describe a successful application of reinforcement learning to autonomous helicopter flight. We first fit a stochastic, nonlinear model of the helicopter dynamics. We then use the model to learn to hover in place, and to fly a number of maneuvers taken from an RC helicopter competition.

## 1 Introduction

Helicopters represent a challenging control problem with high-dimensional, complex, asymmetric, noisy, non-linear, dynamics, and are widely regarded as significantly more difficult to control than fixed-wing aircraft. [7] Consider, for instance, the problem of designing a helicopter that hovers in place. We begin with a single, horizontally-oriented main rotor attached to the helicopter via the rotor shaft. Suppose the main rotor rotates clockwise (viewed from above), blowing air downwards and hence generating upward thrust. By applying clockwise torque to the main rotor to make it rotate, our helicopter experiences an anti-torque that tends to cause the main chassis to spin anti-clockwise. Thus, in the invention of the helicopter, it was necessary to add a tail rotor, which blows air sideways/rightwards to generate an appropriate moment to counteract the spin. But, this sideways force now causes the helicopter to drift leftwards. So, for a helicopter to hover in place, it must actually be tilted slightly to the right, so that the main rotor's thrust is directed downwards and slightly to the left, to counteract this tendency to drift sideways.

The history of helicopters is rife with such tales of ingenious solutions to problems caused by solutions to other problems, and of complex, nonintuitive dynamics that make helicopters challenging to control. In this paper, we describe the successful application of reinforcement learning to designing a controller for autonomous helicopter flight. Due to space constraints, our description of this work is necessarily brief; a detailed treatment is provided in [9]. For a discussion of related work on autonomous flight, also see [9, 12].

## 2 Autonomous Helicopter

The helicopter used in this work was a Yamaha R-50 helicopter, which is approximately 3.6m long, carries up to a 20kg payload, and is shown in Figure 1a. A detailed description of the design and construction of its instrumentation is in [12]. The helicopter carries an Inertial Navigation System (INS) consisting of 3 accelerometers and 3 rate gyroscopes installed in exactly orthogonal x,y,z directions, and a differential GPS system, which with the assistance of a ground station, gives position estimates with a resolution of 2cm. An onboard navigation computer runs a Kalman filter which integrates the sensor information from the GPS, INS, and a digital compass, and reports (at 50Hz) 12 numbers corresponding to the estimates of the helicopter's position $(x, y, z)$, orientation (roll $\phi$, pitch $\theta$, yaw $\omega$), velocity $(\dot{x}, \dot{y}, \dot{z})$ and angular velocities $(\dot{\phi}, \dot{\theta}, \dot{\omega})$.

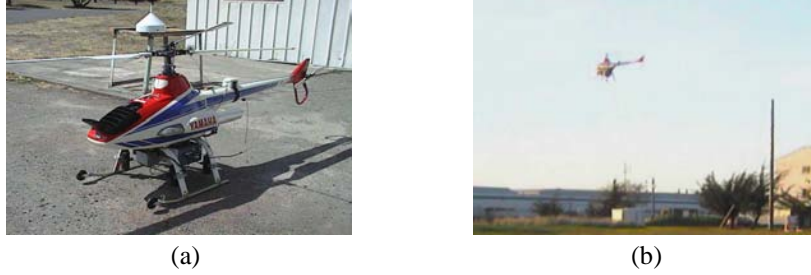

| (a) | (b) |

Figure 1: (a) Autonomous helicopter. (b) Helicopter hovering under control of learned policy.

Most Helicopters are controlled via a 4-dimensional action space:

- $a_1, a_2$: The longtitudinal (front-back) and latitudinal (left-right) cyclic pitch controls. The *rotor plane* is the plane in which the helicopter's rotors rotate. By tilting this plane either forwards/backwards or sideways, these controls cause the helicopter to accelerate forward/backwards or sideways.
- $a_3$: The (main rotor) collective pitch control. As the helicopter main-rotor's blades sweep through the air, they generate an amount of upward thrust that (generally) increases with the angle at which the rotor blades are tilted. By varying the tilt angle of the rotor blades, the collective pitch control affects the main rotor's thrust.
- $a_4$: The tail rotor collective pitch control. Using a mechanism similar to the main rotor collective pitch control, this controls the tail rotor's thrust.

Using the position estimates given by the Kalman filter, our task is to pick good control actions every 50th of a second.

## 3   Model identification

To fit a model of the helicopter's dynamics, we began by asking a human pilot to fly the helicopter for several minutes, and recorded the 12-dimensional helicopter state and 4-dimensional helicopter control inputs as it was flown. In what follows, we used 339 seconds of flight data for model fitting, and another 140 seconds of data for hold-out testing.

There are many natural symmetries in helicopter flight. For instance, a helicopter at (0,0,0) facing east behaves in a way related only by a translation and rotation to one at (10,10,50) facing north, if we command each to accelerate forwards. We would like to encode these symmetries directly into the model rather force an algorithm to learn them from scratch. Thus, model identification is typically done not in the spatial (world) coordinates $s = [x, y, z, \phi, \theta, \omega, \dot{x}, \dot{y}, \dot{z}, \dot{\phi}, \dot{\theta}, \dot{\omega}]$, but instead in the helicopter body coordinates, in which the $x$, $y$, and $z$ axes are forwards, sideways, and down relative to the current position of the helicopter. Where there is risk of confusion, we will use superscript $s$ and $b$ to distinguish between spatial and body coordinates; thus, $\dot{x}^b$ is forward velocity, regardless of orientation. Our model is identified in the body coordinates $s^b = [\phi, \theta, \dot{x}^b, \dot{y}^b, \dot{z}^b, \dot{\phi}, \dot{\theta}, \dot{\omega}]$. which has four fewer variables than $s^s$. Note that once this model is built, it is easily converted back using simple geometry to one in terms of spatial coordinates.

Our main tool for model fitting was locally weighted linear regression (e.g., [11, 3]). Given a dataset $\{(x_i, y_i)\}_{i=1}^m$ where the $x_i$'s are vector-valued inputs and the $y_i$'s are the real-valued outputs to be predicted, we let $X$ be the design matrix whose $i$-th row is $x_i$, and let $y$ be the vector of $y_i$'s. In response to a query at $x$, locally weighted linear regression makes the prediction $y = \beta^T x$, where $\beta = (X^T W X)^{-1} X^T W y$, and $W$ is a diagonal matrix with (say) $W_{ii} = \exp(-\frac{1}{2}(x - x_i)^T \Sigma^{-1}(x - x_i))$, so that the regression gives datapoints near $x$ a larger weight. Here, $\Sigma^{-1}$ determines how weights fall off with distance from $x$, and was picked in our experiments via leave-one-out cross validation.[1] Using the estimator for noise $\sigma^2$ given in [3], this gives a model $y = \beta^T x + \eta$, where $\eta \sim \text{Normal}(0, \sigma^2)$. By

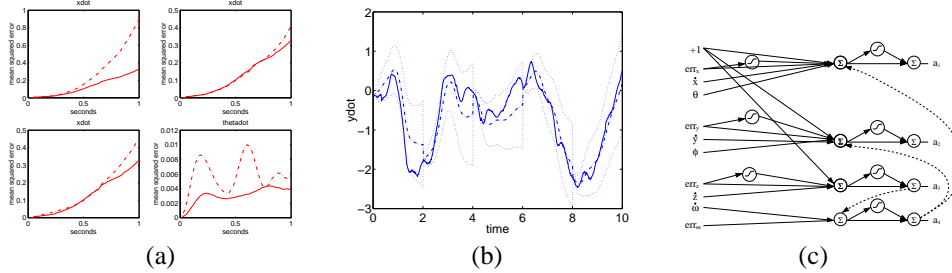

(a)                (b)                (c)

Figure 2: (a) Examples of plots comparing a model fit using the parameterization described in the text (solid lines) to some other models (dash-dot lines). Each point plotted shows the mean-squared error between the predicted value of a state variable—when a model is used to the simulate the helicopter's dynamics for a certain duration indicated on the $x$-axis—and the true value of that state variable (as measured on test data) after the same duration. Top left: Comparison of $\dot{x}$-error to model not using $a_{1s}$, etc. terms. Top right: Comparison of $\dot{x}$-error to model omitting intercept (bias) term. Bottom: Comparison of $\dot{x}$ and $\dot{\theta}$ to linear deterministic model identified by [12]. (b) The solid line is the true helicopter $\dot{y}$ state on 10s of test data. The dash-dot line is the helicopter state predicted by our model, given the initial state at time 0 and all the intermediate control inputs. The dotted lines show two standard deviations in the estimated state. Every two seconds, the estimated state is "reset" to the true state, and the track restarts with zero error. Note that the estimated state is of the full, high-dimensional state of the helicopter, but only $\dot{y}$ is shown here. (c) Policy class. The picture inside the circles indicate whether a node outputs the sum of their inputs, or the $\tanh$ of the sum of their inputs. Each edge with an arrow in the picture denotes a tunable parameter. The solid lines show the hovering policy class (Section 5). The dashed lines show the extra weights added for trajectory following (Section 6).

applying locally-weighted regression with the state $s_t$ and action $a_t$ as inputs, and the one-step differences (e.g., $\phi_{t+1} - \phi_t$) of each of the state variables in turn as the target output, this gives us a non-linear, stochastic, model of the dynamics, allowing us to predict $s_{t+1}$ as a function of $s_t$ and $a_t$ plus noise.

We actually used several refinements to this model. Similar to the use of body coordinates to exploit symmetries, there is other prior knowledge that can be incorporated. Since both $\phi_t$ and $\dot{\phi}_t$ are state variables, and we know that (at 50Hz) $\phi_{t+1} \approx \phi_t + \dot{\phi}_t/50$, there is no need to carry out a regression for $\phi$. Similarly, we know that the roll angle $\phi$ of the helicopter should have no direct effect on forward velocity $\dot{x}$. So, when performing regression to estimate $\dot{x}$, the coefficient in $\beta$ corresponding to $\phi$ can be set to 0. This allows us to reduce the number of parameters that have to be fit. Similar reasoning allows us to conclude (cf. [12]) that certain other parameters should be 0, $1/50$ or $g$ (gravity), and these were also hard-coded into the model. Finally, we added three extra (unobserved) variables $a_{1s}, b_{1s}, \dot{\omega}_{fb}$ to model latencies in the responses to the controls. (See [9] for details.)

Some of the (other) choices that we considered in selecting a model include whether to use the $a_{1s}$, $b_{1s}$ and/or $\dot{\omega}_{fb}$ terms; whether to include an intercept term; at what frequency to identify the model; whether to hardwire certain coefficients as described; and whether to use weighted or unweighted linear regression. Our main tool for choosing among the models was plots such as those shown in Figure 2a. (See figure caption.) We were particularly interested in checking how accurate a model is not just for predicting $s_{t+1}$ from $s_t, a_t$, but how accurate it is at longer time scales. Each of the panels in Figure 2a shows, for a model, the mean-squared error (as measured on test data) between the helicopter's true position and the estimated position at a certain time in the future (indicated on the $x$-axis).

The helicopter's blade-tip moves at an appreciable fraction of the speed of sound. Given the

---

and the presence of temporally close-by samples—which will be spatially close-by as well—may make data seem more abundant than in reality (leading to bigger $\Sigma^{-1}$ than might be optimal for test data). Thus, when leaving out a sample in cross validation, we actually left out a large window (16 seconds) of data around that sample, to diminish this bias.

danger and expense (about \$70,000) of autonomous helicopters, we wanted to verify the fitted model carefully, so as to be reasonably confident that a controller tested successfully in simulation will also be safe in real life. Space precludes a full discussion, but one of our concerns was the possibility that unmodeled correlations in $\eta$ might mean the noise variance of the actual dynamics is much larger than predicted by the model. (See [9] for details.) To check against this, we examined many plots such as shown in Figure 2, to check that the helicopter state "rarely" goes outside the errorbars predicted by our model *at various time scales* (see caption).

## 4  Reinforcement learning: The PEGASUS algorithm

We used the PEGASUS reinforcement learning algorithm of [10], which we briefly review here. Consider an MDP with state space $S$, initial state $s_0 \in S$, action space $A$, state transition probabilities $P_{sa}(\cdot)$, reward function $R : S \mapsto \mathbb{R}$, and discount $\gamma$. Also let some family $\Pi$ of policies $\pi : S \mapsto A$ be given, and suppose our goal is to find a policy in $\Pi$ with high utility, where the policy of $\pi$ is defined to be

$$U(\pi) = \mathrm{E}[R(s_0) + \gamma R(s_1) + \gamma^2 R(s_2) + \cdots | \pi],$$

where the expectation is over the random sequence of states $s_0, s_1, \ldots$ visited over time when $\pi$ is executed in the MDP starting from state $s_0$.

These utilities are in general intractable to calculate exactly, but suppose we have a computer simulator of the MDP's dynamics—that is, a program that inputs $s, a$ and outputs $s'$ drawn from $P_{sa}(\cdot)$. Then a standard way to define an estimate $\hat{U}(\pi)$ of $U(\pi)$ is via Monte Carlo: We can use the simulator to sample a trajectory $s_0, s_1, \ldots$, and by taking the empirical sum of discounted rewards $R(s_0) + \gamma R(s_1) + \cdots$ on this sequence, we obtain one "sample" with which to estimate $U(\pi)$. More generally, we could generate $m$ such sequences, and average to obtain a better estimator. We can then try to optimize the estimated utilities and search for "$\arg\max_\pi \hat{U}(\pi)$."

Unfortunately, this is a difficult stochastic optimization problem: Evaluating $\hat{U}(\pi)$ involves a Monte Carlo sampling process, and two different evaluations of $\hat{U}(\pi)$ will typically give slightly different answers. Moreover, even if the number of samples $m$ that we average over is arbitrarily large, $\hat{U}(\pi)$ will fail with probability 1 to be a ("uniformly") good estimate of $U(\pi)$. In our experiments, this fails to learn any reasonable controller for our helicopter.

The PEGASUS method uses the observation that almost all computer simulations of the form described sample $s' \sim P_{sa}(\cdot)$ by first calling a random number generator to get one (or more) random numbers $p$, and then calculating $s'$ as some deterministic function of the input $s, a$ and the random $p$. If we demand that the simulator expose its interface to the random number generator, then by pre-sampling all the random numbers $p$ in advance and fixing them, we can then use these same, fixed, random numbers to evaluate *any* policy. Since all the random numbers are fixed, $\hat{U} : \Pi \mapsto \mathbb{R}$ is just an ordinary deterministic function, and standard search heuristics can be used to search for $\arg\max_\pi \hat{U}(\pi)$. Importantly, this also allows us to show that, so long as we average over a number of samples $m$ that is at most polynomial in all quantities of interest, then with high probability, $\hat{U}$ will be a uniformly good estimate of $U$ ($|\hat{U}(\pi) - U(\pi)| \le \epsilon$). This also allows us to give guarantees on the performance of the solutions found. For further discussion of PEGASUS and other work such as the variance reduction and gradient estimation methods (cf. [6, 5]), see [9].

## 5  Learning to Hover

One previous attempt had been made to use a learning algorithm to fly this helicopter, using $\mu$-synthesis [2]. This succeeded in flying the helicopter in simulation, but not on the actual helicopter (Shim, pers. comm.). Similarly, preliminary experiments using $H_2$ and $H_\infty$ controllers to fly a similar helicopter were also unsuccessful. These comments should not be taken as conclusive of the viability of any of these methods; rather, we take them to be indicative of the difficulty and subtlety involved in learning a helicopter controller.

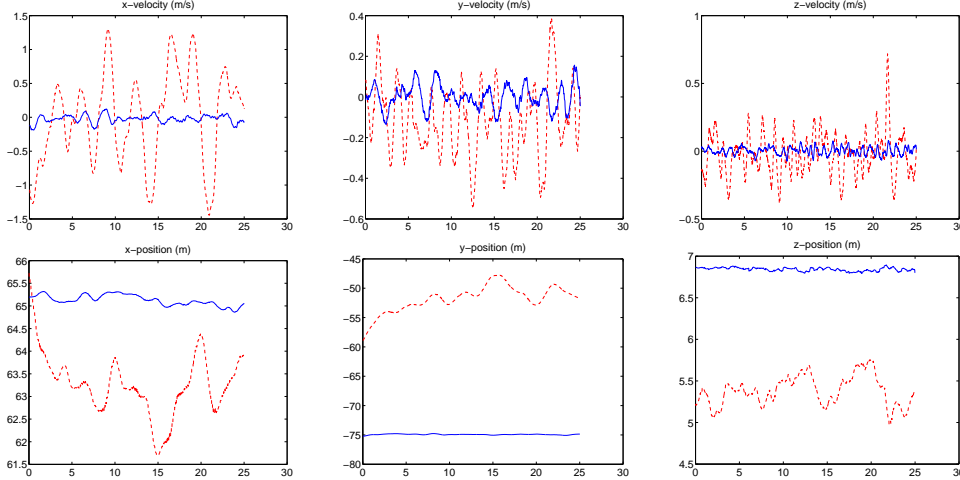

Figure 3: Comparison of hovering performance of learned controller (solid line) vs. Yamaha licensed/specially trained human pilot (dotted line). Top: $x, y, z$ velocities. Bottom: $x, y, z$ positions.

We began by learning a policy for hovering in place. We want a controller that, given the current helicopter state and a desired hovering position and orientation $(x^*, y^*, z^*, \omega^*)$, computes controls $a \in [-1, 1]^4$ to make it hover stably there. For our policy class $\Pi$, we chose the simple neural network depicted in Figure 2c (solid edges only). Each of the edges in the figure represents a weight, and the connections were chosen via simple reasoning about which control channel should be used to control which state variables. For instance, consider the the longitudinal (forward/backward) cyclic pitch control $a_1$, which causes the rotor plane to tilt forward/backward, thus causing the helicopter to pitch (and/or accelerate) forward or backward. From Figure 2c, we can read off the $a_1$ control control as

$$t_1 = w_1 + w_2 \operatorname{err}_{x^b} + w_3 \tanh(w_4 \operatorname{err}_{x^b}) + w_5 \dot{x}^b + w_6 \theta; \ a_1 = w_7 \tanh(w_8 t_1) + w_9 t_1.$$

Here, the $w_i$'s are the tunable parameters (weights) of the network, and $\operatorname{err}_{x^b} = x^b - x^b_{\text{desired}}$ is defined to be the error in the $x^b$-position (forward direction, in body coordinates) between where the helicopter currently is and where we wish it to hover.

We chose a quadratic cost function on the (spatial representation of the) state, where[2]

$$R(s) = -(\alpha_x(x - x^*)^2 + \alpha_y(y - y^*)^2 + \alpha_z(y - y^*)^2 + \alpha_{\dot{x}}\dot{x}^2 + \alpha_{\dot{y}}\dot{y}^2 + \alpha_{\dot{z}}\dot{z}^2 + \alpha_\omega(\omega - \omega^*)^2). \quad (1)$$

This encourages the helicopter to hover near $(x^*, y^*, z^*, \omega^*)$, while also keeping the velocity small and not making abrupt movements. The weights $\alpha_x, \alpha_y$, etc. (distinct from the weights $w_i$ parameterizing our policy class) were chosen to scale each of the terms to be roughly the same order of magnitude. To encourage small actions and smooth control of the helicopter, we also used a quadratic penalty for actions: $R(a) = -(\alpha_{a_1}a_1^2 + \alpha_{a_2}a_2^2 + \alpha_{a_3}a_3^2 + \alpha_{a_4}a_4^2)$, and the overall reward was $R(s, a) = R(s) + R(a)$.

Using the model identified in Section 3, we can now apply PEGASUS to define approximations $\hat{U}(\pi)$ to the utilities of policies. Since policies are smoothly parameterized in the weights, and the dynamics are themselves continuous in the actions, the estimates of utilities are also continuous in the weights.[3] We may thus apply standard hillclimbing algorithms to maximize $\hat{U}(\pi)$ in terms of the policy's weights. We tried both a gradient

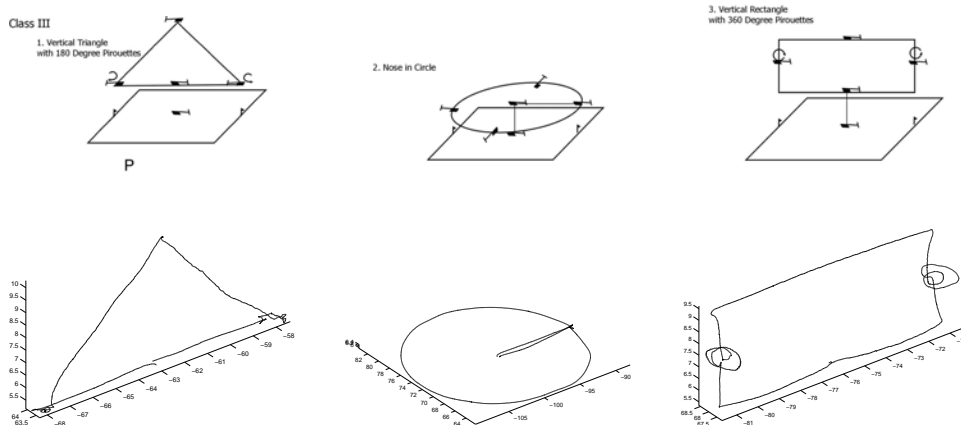

Figure 4: Top row: Maneuver diagrams from RC helicopter competition. [Source: www.modelaircraft.org]. Bottom row: Actual trajectories flown using learned controller.

ascent algorithm, in which we numerically evaluate the derivative of $\hat{U}(\pi)$ with respect to the weights and then take a step in the indicated direction, and a random-walk algorithm in which we propose a random perturbation to the weights, and move there if it increases $\hat{U}(\pi)$. Both of these algorithms worked well, though with gradient ascent, it was important to scale the derivatives appropriately, since the estimates of the derivatives were sometimes numerically unstable.[4] It was also important to apply some standard heuristics to prevent its solutions from diverging (such as verifying after each step that we did indeed take a step uphill on the objective $\hat{U}$, and undoing/redoing the step using a smaller stepsize if this was not the case).

The most expensive step in policy search was the repeated Monte Carlo evaluation to obtain $\hat{U}(\pi)$. To speed this up, we parallelized our implementation, and Monte Carlo evaluations using different samples were run on different computers, and the results were then aggregated to obtain $\hat{U}(\pi)$. We ran PEGASUS using 30 Monte Carlo evaluations of 35 seconds of flying time each, and $\gamma = 0.9995$. Figure 1b shows the result of implementing and running the resulting policy on the helicopter. On its maiden flight, our learned policy was successful in keeping the helicopter stabilized in the air. (We note that [1] was also successful at using our PEGASUS algorithm to control a subset, the cyclic pitch controls, of a helicopter's dynamics.)

We also compare the performance of our learned policy against that of our human pilot trained and licensed by Yamaha to fly the R-50 helicopter. Figure 5 shows the velocities and positions of the helicopter under our learned policy and under the human pilot's control. As we see, our controller was able to keep the helicopter flying more stably than was a human pilot. Videos of the helicopter flying are available at

$$\texttt{http://www.cs.stanford.edu/\textasciitilde ang/nips03/}$$

## 6 Flying competition maneuvers

We were next interested in making the helicopter learn to fly several challenging maneuvers. The Academy of Model Aeronautics (AMA) (to our knowledge the largest RC helicopter organization) holds an annual RC helicopter competition, in which helicopters have to be accurately flown through a number of maneuvers. This competition is organized into Class I (for beginners, with the easiest maneuvers) through Class III (with the most difficult maneuvers, for the most advanced pilots). We took the first three maneuvers from the most challenging, Class III, segment of their competition.

Figure 4 shows maneuver diagrams from the AMA web site. In the first of these maneuvers

(III.1), the helicopter starts from the middle of the base of a triangle, flies backwards to the lower-right corner, performs a $180°$ pirouette (turning in place), flies backwards up an edge of the triangle, backwards down the other edge, performs another $180°$ pirouette, and flies backwards to its starting position. Flying backwards is a significantly less stable maneuver than flying forwards, which makes this maneuver interesting and challenging. In the second maneuver (III.2), the helicopter has to perform a nose-in turn, in which it flies backwards out to the edge of a circle, pauses, and then flies in a circle but always keeping the nose of the helicopter pointed at center of rotation. After it finishes circling, it returns to the starting point. Many human pilots seem to find this second maneuver particularly challenging. Lastly, maneuver III.3 involves flying the helicopter in a vertical rectangle, with two $360°$ pirouettes in opposite directions halfway along the rectangle's vertical segments.

How does one design a controller for flying trajectories? Given a controller for keeping a system's state at a point $(x^*, y^*, z^*, \omega^*)$, one standard way to make the system move through a particular trajectory is to slowly vary $(x^*, y^*, z^*, \omega^*)$ along a sequence of set points on that trajectory. (E.g., see [4].) For instance, if we ask our helicopter to hover at $(0, 0, 0, 0)$, then a fraction of a second later ask it to hover at $(0.01, 0, 0, 0)$, then at $(0.02, 0, 0, 0)$ and so on, our helicopter will slowly fly in the $x^s$-direction. By taking this procedure and "wrapping" it around our old policy class from Figure 2c, we thus obtain a computer program—that is, a new policy class—not just for hovering, but also for flying *arbitrary* trajectories. I.e., we now have a family of policies that take as input a *trajectory*, and that attempt to make the helicopter fly that trajectory. Moreover, we can now also *retrain the policy's parameters for accurate trajectory following*, not just hovering.

Since we are now flying trajectories and not only hovering, we also augmented the policy class to take into account more of the coupling between the helicopter's different sub-dynamics. For instance, the simplest way to turn is to change the tail rotor collective pitch/thrust, so that it yaws either left or right. This works well for small turns, but for large turns, the thrust from the tail rotor also tends to cause the helicopter to drift sideways. Thus, we enriched the policy class to allow it to correct for this drift by applying the appropriate cyclic pitch controls. Also, having a helicopter climb or descend changes the amount of work done by the main rotor, and hence the amount of torque/anti-torque generated, which can cause the helicopter to turn. So, we also added a link between the collective pitch control and the tail rotor control. These modifications are shown in Figure 2c (dashed lines).

We also needed to specify a reward function for trajectory following. One simple choice for $R$ would have been to use Equation (1) with the newly-defined (time-varying) $(x^*, y^*, z^*, \omega^*)$. But we did not consider this to be a good choice. Specifically, consider making the helicopter fly in the increasing $x$-direction, so that $(x^*, y^*, z^*, \omega^*)$ starts off as $(0, 0, 0, 0)$ (say), and has its first coordinate $x^*$ slowly increased over time. Then, while the actual helicopter position $x^s$ will indeed increase, it will also almost certainly lag consistently behind $x^*$. This is because the hovering controller is always trying to "catch up" to the moving $(x^*, y^*, z^*, \omega^*)$. Thus, $x - x^*$ may remain large, and the helicopter will continuously incur a $x - x^*$ cost, even if it is in fact flying a very accurate trajectory in the increasing $x$-direction exactly as desired. It would be undesirable to have the helicopter risk trying to fly more aggressively to reduce this fake "error," particularly if it is at the cost of increased error in the other coordinates. So, we changed the reward function to penalize deviation not from $(x^*, y^*, z^*, \omega^*)$, but instead deviation from $(x_p, y_p, z_p, \omega_p)$, where $(x_p, y_p, z_p, \omega_p)$ is the "projection" of the helicopter's position onto the path of the idealized, desired trajectory. (In our example of flying in a straight line, for a helicopter at $(x, y, z, \omega)$, we easily see $(x_p, y_p, z_p, \omega_p) = (x, 0, 0, 0)$.) Thus, we imagine an "external observer" that looks at the actual helicopter state and estimates which part of the idealized trajectory the helicopter is trying to fly through (taking care not to be confused if a trajectory loops back on itself), and the learning algorithm pays a penalty that is quadratic between the actual position and the "tracked" position on the idealized trajectory.

We also needed to make sure the helicopter is rewarded for making progress along the

trajectory. To do this, we used the potential-based shaping rewards of [8]. Since, we are already tracking where along the desired trajectory the helicopter is, we chose a potential function that increases along the trajectory. Thus, whenever the helicopter's $(x_p, y_p, z_p, \omega_p)$ makes forward progress along this trajectory, it receives positive reward. (See [8].)

Finally, our modifications have decoupled our definition of the reward function from $(x^*, y^*, z^*, \omega^*)$ and the evolution of $(x^*, y^*, z^*, \omega^*)$ in time. So, we are now also free to consider allowing $(x^*, y^*, z^*, \omega^*)$ to evolve in a way that is *different* from the path of the desired trajectory, but nonetheless in way that allows the helicopter to follow the actual, desired trajectory more accurately. (In control theory, there is a related practice of using the inverse dynamics to obtain better tracking behavior.) We considered several alternatives, but the main one used ended up being a modification for flying trajectories that have both a vertical and a horizontal component (such as along the two upper edges of the triangle in III.1). Specifically, it turns out that the $z$ (vertical)-response of the helicopter is very fast: To climb, we need only increase the collective pitch control, which almost immediately causes the helicopter to start accelerating upwards. In contrast, the $x$ and $y$ responses are much slower. Thus, if $(x^*, y^*, z^*, \omega^*)$ moves at $45°$ upwards as in maneuver III.1, the helicopter will tend to track the $z$-component of the trajectory much more quickly, so that it accelerates into a climb steeper than $45°$, resulting in a "bowed-out" trajectory. Similarly, an angled descent results in a "bowed-in" trajectory. To correct for this, we artificially slowed down the $z$-response, so that when $(x^*, y^*, z^*, \omega^*)$ is moving into an angled climb or descent, the $(x^*, y^*, \omega^*)$ portion will evolve normally with time, but the changes to $z^*$ will be delayed by $t$ seconds, where $t$ here is another parameter in our policy class, to be automatically learned by our algorithm.

Using this setup and retraining our policy class' parameters for accurate trajectory following, we were able to learn a policy that flies all three of the competition maneuvers fairly accurately. Figure 4 (bottom) shows actual trajectories taken by the helicopter while flying these maneuvers. Videos of the helicopter flying these maneuvers are also available at the URL given at the end of Section 5.

## Footnotes

[1]Actually, since we were fitting a model to a time-series, samples tend to be correlated in time,

[2]The $\omega - \omega^*$ error term is computed with appropriate wrapping about $2\pi$ rad, so that if $\omega^* = 0.01$ rad, and the helicopter is currently facing $\omega = 2\pi - 0.01$ rad, the error is 0.02, not $2\pi - 0.02$ rad.

[3]Actually, this is not true. One last component of the reward that we did not mention earlier was that, if in performing the locally weighted regression, the matrix $X^T W X$ is singular to numerical precision, then we declare the helicopter to have "crashed," terminate the simulation, and give it a huge negative (-50000) reward. Because the test checking if $X^T W X$ is singular to numerical precision returns either 1 or 0, $\hat{U}(\pi)$ has a discontinuity between "crash" and "not-crash."

[4]A problem exacerbated by the discontinuities described in the previous footnote.

## References

[1] J. Bagnell and J. Schneider. Autonomous helicopter control using reinforcement learning policy search methods. In *Int'l Conf. Robotics and Automation*. IEEE, 2001.

[2] G. Balas, J. Doyle, K. Glover, A. Packard, and R. Smith. $\mu$-analysis and synthesis toolbox user's guide, 1995.

[3] W. Cleveland. Robust locally weighted regression and smoothing scatterplots. *J. Amer. Stat. Assoc*, 74, 1979.

[4] Gene F. Franklin, J. David Powell, and Abbas Emani-Naeini. *Feedback Control of Dynamic Systems*. Addison-Wesley, 1995.

[5] Y. Ho and X. Cao. *Pertubation analysis of discrete event dynamic systems*. Kluwer, 1991.

[6] J. Kiefer and J. Wolfowitz. Stochastic estimation of the maximum of a regression function. *Annals of Mathematical Statistics*, 23:462–466, 1952.

[7] J. Leishman. *Principles of Helicopter Aerodynamics*. Cambridge Univ. Press, 2000.

[8] A. Y. Ng, D. Harada, and S. Russell. Policy invariance under reward transformations: Theory and application to reward shaping. In *Proc. 16th ICML*, pages 278–287, 1999.

[9] Andrew Y. Ng. *Shaping and policy search in reinforcement learning*. PhD thesis, EECS, University of California, Berkeley, 2003.

[10] Andrew Y. Ng and Michael I. Jordan. PEGASUS: A policy search method for large MDPs and POMDPs. In *Proc. 16th Conf. Uncertainty in Artificial Intelligence*, 2000.

[11] C. Atkeson S. Schaal and A. Moore. Locally weighted learning. *AI Review*, 11, 1997.

[12] Hyunchul Shim. *Hierarchical flight control system synthesis for rotorcraft-based unmanned aerial vehicles*. PhD thesis, Mech. Engr., U.C. Berkeley, 2000.
